# Error Bounds for Transductive Learning via Compression and Clustering

**Philip Derbeko**      **Ran El-Yaniv**      **Ron Meir**

Technion - Israel Institute of Technology

{philip,rani}@cs.technion.ac.il   rmeir@ee.technion.ac.il

## Abstract

This paper is concerned with transductive learning. Although transduction appears to be an easier task than induction, there have not been many provably useful algorithms and bounds for transduction. We present explicit error bounds for transduction and derive a general technique for devising bounds within this setting. The technique is applied to derive error bounds for compression schemes such as (transductive) SVMs and for transduction algorithms based on clustering.

## 1   Introduction and Related Work

In contrast to inductive learning, in the *transductive setting* the learner is given both the training and test sets prior to learning. The goal of the learner is to infer (or "transduce") the labels of the test points. The transduction setting was introduced by Vapnik [1, 2] who proposed basic bounds and an algorithm for this setting. Clearly, inferring the labels of points in the test set can be done using an inductive scheme. However, as pointed out in [2], it makes little sense to solve an easier problem by 'reducing' it to a much more difficult one. In particular, the prior knowledge carried by the (unlabeled) test points can be incorporated into an algorithm, potentially leading to superior performance. Indeed, a number of papers have demonstrated empirically that transduction can offer substantial advantage over induction whenever the training set is small or moderate (see e.g. [3, 4, 5, 6]). However, unlike the current state of affairs in induction, the question of what are provably effective learning *principles* for transduction is quite far from being resolved.

In this paper we provide new error bounds and a general technique for transductive learning. Our technique is based on bounds that can be viewed as an extension of McAllester's PAC-Bayesian framework [7, 8] to transductive learning. The main advantage of using this framework in transduction is that here priors can be selected after observing the unlabeled data (but before observing the labeled sample). This flexibility allows for the choice of "compact priors" (with small support) and therefore, for tight bounds. Another simple observation is that the PAC-Bayesian framework can be operated with polynomially (in $m$, the training sample size) many different priors simultaneously. Altogether, this added flexibility, of using *data-dependent multiple priors* allows for easy derivation of tight error bounds for "compression schemes" such as (transductive) SVMs and for clustering algorithms.

We briefly review some previous results. The idea of transduction, and a specific algorithm for SVM transductive learning, was introduced and studied by Vapnik (e.g. [2]), where an

error bound is also proposed. However, this bound is implicit and rather unwieldy and, to the best of our knowledge, has not been applied in practical situations. A PAC-Bayes bound [7] for transduction with Perceptron Decision Trees is given in [9]. The bound is data-dependent depending on the number of decision nodes, the margins at each node and the sample size. However, the authors state that the transduction bound is not much tighter than the induction bound. Empirical tests show that this transduction algorithm performs slightly better than induction in terms of the test error, however, the advantage is usually statistically insignificant. Refining the algorithm of [2] a transductive algorithm based on a SVMs is proposed in [3]. The paper also provides empirical tests indicating that transduction is advantageous in the text categorization domain. An error bound for transduction, based on the *effective* VC Dimension, is given in [10]. More recently Lanckriet *et al.* [11] derived a transductive bound for kernel methods based on spectral properties of the kernel matrix. Blum and Langford [12] recently also established an implicit bound for transduction, in the spirit of the results in [2].

## 2 The Transduction Setup

We consider the following setting proposed by Vapnik ([2] Chp. 8), which for simplicity is described in the context of binary classification (the general case will be discussed in the full paper). Let $\mathcal{H}$ be a set of binary hypotheses consisting of functions from input space $\mathcal{X}$ to $\{\pm 1\}$ and let $X_{m+u} = \{x_1, \ldots, x_{m+u}\}$ be a set of points from $\mathcal{X}$ each of which is chosen i.i.d. according to some unknown distribution $\mu(x)$. We call $X_{m+u}$ the *full sample*. Let $X_m = \{x_1, \ldots, x_m\}$ and $Y_m = \{y_1, \ldots, y_m\}$, where $X_m$ is drawn uniformly from $X_{m+u}$ and $y_i \in \{\pm 1\}$. The set $S_m = \{(x_1, y_1), \ldots, (x_m, y_m)\}$ is referred to as a *training sample*. In this paper we assume that $y_i = \phi(x_i)$ for some unknown function $\phi$. The remaining subset $X_u = X_{m+u} \setminus X_m$ is referred to as the *unlabeled sample*. Based on $S_m$ and $X_u$ our goal is to choose $h \in \mathcal{H}$ which predicts the labels of points in $X_u$ as accurately as possible. For each $h \in \mathcal{H}$ and a set $Z = x_1, \ldots, x_{|Z|}$ of samples define

$$R_h(Z) = \frac{1}{|Z|} \sum_{i=1}^{|Z|} \ell(h(x_i), y_i), \tag{1}$$

where in our case $\ell(\cdot, \cdot)$ is the zero-one loss function. Our goal in transduction is to learn an $h$ such that $R_h(X_u)$ is as small as possible. This problem setup is summarized by the following transduction "protocol" introduced in [2] and referred to as ***Setting 1***:

(i) A *full sample* $X_{m+u} = \{x_1, \ldots, x_{m+u}\}$ consisting of arbitrary $m + u$ points is given.[1]

(ii) We then choose uniformly at random the training sample $X_m \subseteq X_{m+u}$ and receive its labeling $Y_m$; the resulting *training set* is $S_m = (X_m, Y_m)$ and the remaining set $X_u$ is the *unlabeled sample*, $X_u = X_{m+u} \setminus X_m$;

(iii) Using both $S_m$ and $X_u$ we select a classifier $h \in \mathcal{H}$ whose quality is measured by $R_h(X_u)$.

Vapnik [2] also considers another formulation of transduction, referred to as ***Setting 2***:

(i) We are given a training set $S_m = (X_m, Y_m)$ selected i.i.d according to $\mu(x, y)$.

(ii) An independent test set $S_u = (X_u, Y_u)$ of $u$ samples is then selected in the same manner.

(iii) We are required to choose our best $h \in \mathcal{H}$ based on $S_m$ and $X_u$ so as to minimize

$$R_{m,u}(h) = \int \frac{1}{u} \sum_{i=m+1}^{m+u} \ell\left(h(x_i), y_i\right) d\mu(x_1, y_1) \cdots d\mu(x_{m+u}, y_{m+u}). \quad (2)$$

Even though Setting 2 may appear more applicable in practical situations than Setting 1, the derivation of theoretical results can be easier within Setting 1. Nevertheless, as far as the expected losses are concerned, Vapnik [2] shows that an error bound in Setting 1 implies an equivalent bound in Setting 2. In view of this result we restrict ourselves in the sequel to Setting 1.

We make use of the following quantities, which are all instances of (1). The quantity $R_h(X_{m+u})$ is called the *full sample risk* of the hypothesis $h$, $R_h(X_u)$ is referred to as the *transduction risk* (of $h$), and $R_h(X_m)$ is the *training error* (of $h$). Thus, $R_h(X_m)$ is the standard training error denoted by $\hat{R}_h(S_m)$. While our objective in transduction is to achieve small error over the unlabeled set (i.e. to minimize $R_h(X_u)$), it turns out that it is much easier to derive error bounds for the full sample risk. The following simple lemma translates an error bound on $R_h(X_{m+u})$, the full sample risk, to an error bound on the transduction risk $R_h(X_u)$.

**Lemma 2.1** *For any $h \in \mathcal{H}$ and any $C$*

$$R_h(X_{m+u}) \leq \hat{R}_h(S_m) + C \qquad \Leftrightarrow \qquad R_h(X_u) \leq \hat{R}_h(S_m) + \frac{m+u}{u} \cdot C. \quad (3)$$

**Proof:** For any $h$

$$R_h(X_{m+u}) = \frac{mR_h(X_m) + uR_h(X_u)}{m+u}. \quad (4)$$

Substituting $\hat{R}_h(S_m)$ for $R_h(X_m)$ in (4) and then substituting the result for the left-hand side of (3) we get

$$R_h(X_{m+u}) = \frac{m\hat{R}_h(S_m) + uR_h(X_u)}{m+u} \leq \hat{R}_h(S_m) + C.$$

The equivalence (3) is now obtained by isolating $R_h(X_u)$ on the left-hand side. $\qquad \square$

## 3  General Error Bounds for Transduction

Consider a hypothesis class $\mathcal{H}$ and assume for simplicity that $\mathcal{H}$ is countable; in fact, in the case of transduction it suffices to consider a finite hypothesis class. To see this note that all $m + u$ points are known in advance. Thus, in the case of binary classification (for example) it suffices to consider at most $2^{m+u}$ possible dichotomies. Recall that in the setting considered we select a sub-sample of $m$ points from the set $X_{m+u}$ of cardinality $m+u$. This corresponds to a selection of $m$ points *without* replacement from a set of $m+u$ points, leading to the $m$ points being *dependent*. A naive utilization of large deviation bounds would therefore not be directly applicable in this setting. However, Hoeffding (see Theorem 4 in [13]) pointed out a simple procedure to transform the problem into one involving *independent* data. While this procedure leads to non-trivial bounds, it does not fully take advantage of the transductive setting and will not be used here. Consider for simplicity the case of binary classification. In this case we make use of the following concentration inequality, based on [14].

**Theorem 3.1** *Let $\mathcal{C} = \{c_1, \ldots, c_N\}$, $c_i \in \{0, 1\}$, be a finite set of binary numbers, and set $\bar{c} = (1/N) \sum_{i=1}^{N} c_i$. Let $Z_1, \ldots, Z_m$, be random variables obtaining their values*

*by sampling $\mathcal{C}$ uniformly at random **without** replacement. Set $Z = (1/m)\sum_{i=1}^m Z_i$ and $\beta = m/N$. Then, if $^2$ $\varepsilon \le \min\{1 - \bar{c}, \bar{c}(1-\beta)/\beta\}$,*

$$\mathbf{Pr}\{Z - \mathbf{E}Z > \varepsilon\} \le \exp\left\{-mD(\bar{c} + \varepsilon\|\bar{c}) - (N - m)\, D\left(\bar{c} - \frac{\beta\varepsilon}{1-\beta}\middle\|\bar{c}\right) + 7\log(N+1)\right\},$$

*where $D(p\|q) = p\log(p/q) = (1-p)\log(1-p)/(1-q)$, $p, q, \in [0, 1]$ is the binary Kullback-Leibler divergence.*

Using this result we obtain the following error bound for transductive classification.

**Theorem 3.2** *Let $X_{m+u} = X_m \cup X_u$ be the full sample and let $\mathbf{p} = \mathbf{p}(X_{m+u})$ be a (prior) distribution over the class of binary hypotheses $\mathcal{H}$ that may depend on the full sample. Let $\delta \in (0, 1)$ be given. Then, with probability at least $1 - \delta$ over choices of $S_m$ (from the full sample) the following bound holds for any $h \in \mathcal{H}$,*

$$R_h(X_u) \le \hat{R}_h(S_m) + \sqrt{\left(\frac{2\hat{R}_h(S_m)(m+u)}{u}\right)\frac{\log\frac{1}{\mathbf{p}(h)} + \ln\frac{m}{\delta} + 7\log(m+u+1)}{m-1}}$$

$$+ \frac{2\left(\log\frac{1}{\mathbf{p}(h)} + \ln\frac{m}{\delta} + 7\log(m+u+1)\right)}{m-1}. \tag{5}$$

**Proof:** (sketch) In our transduction setting the set $X_m$ (and therefore $S_m$) is obtained by sampling the full sample $X_{m+u}$ uniformly at random without replacement. We first claim that

$$\mathbf{E}_{\Sigma_m}\hat{R}_h(S_m) = R_h(X_{m+u}), \tag{6}$$

where $\mathbf{E}_{\Sigma_m}(\cdot)$ is the expectation with respect to a random choice of $S_m$ from $X_{m+u}$ without replacement. This is shown as follows.

$$\mathbf{E}_{\Sigma_m}\hat{R}_h(S_m) = \frac{1}{\binom{m+u}{m}}\sum_{S_m}\hat{R}_h(S_m) = \frac{1}{\binom{m+u}{m}}\sum_{X_m \subseteq X_{m+n}}\frac{1}{m}\sum_{x \in S_m}\ell(h(x), \phi(x)).$$

By symmetry, all points $x \in X_{m+u}$ are counted on the right-hand side an equal number of times; this number is precisely $\binom{m+u}{m} - \binom{m+u-1}{m} = \binom{m+u-1}{m-1}$. The equality (6) is obtained by considering the definition of $R_h(X_{m+u})$ and noting that $\binom{m+u-1}{m-1}/\binom{m+u}{m} = \frac{m}{m+u}$.

The remainder of the proof combines Theorem 3.1 and the techniques presented in [15]. The details will be provided in the full paper. $\qquad\square$

Notice that when $\hat{R}_h(S_m) \to 0$ the square root in (5) vanishes and faster rates are obtained. An important feature of Theorem 3.2 is that it allows one to use the sample $X_{m+u}$ in order to choose the prior distribution $\mathbf{p}(h)$. This advantage has already been alluded to in [2], but does not seem to have been widely used in practice. Additionally, observe that (5) holds with probability at least $1 - \delta$ with respect to the random selection of sub-samples of size $m$ from the fixed set $X_{m+u}$. This should be contrasted with the standard inductive setting results where the probabilities are with respect to a random choice of $m$ training points chosen i.i.d. from $\mu(x, y)$.

The next bound we present is analogous to McAllester's Theorem 1 in [8]. This theorem concerns Gibbs *composite classifiers*, which are distributions over the *base classifiers* in $\mathcal{H}$. For any distribution $\mathbf{q}$ over $\mathcal{H}$ denote by $G_{\mathbf{q}}$ the Gibbs classifier, which classifies an

---

$^2$The second condition, $\varepsilon \le \bar{c}(1 - \beta)/\beta$, simply guarantees that the number of 'ones' in the sub-sample does not exceed their number in the original sample.

instance (in $X_u$) by randomly choosing, according to $\mathbf{q}$, one hypothesis $h \in \mathcal{H}$. For Gibbs classifiers we now extend definition (1) as follows. Let $Z = x_1, \ldots, x_{|Z|}$ be any set of samples and let $G_\mathbf{q}$ be a Gibbs classifier over $\mathcal{H}$. The risk of $G_\mathbf{q}$ over $Z$ is $R_{G_\mathbf{q}}(Z) = \mathbf{E}_{h \sim \mathbf{q}} \left\{ (1/|Z|) \sum_{i=1}^{|Z|} \ell(h(x_i), \phi(x_i)) \right\}$. As before, when $Z = X_m$ (the training set) we use the standard notation $\hat{R}_{G_\mathbf{q}}(S_m) = R_{G_\mathbf{q}}(X_m)$. Due to space limitations, the proof of the following theorem will appear in the full paper.

**Theorem 3.3** *Let $X_{m+u}$ be the full sample. Let $\mathbf{p}$ be a distribution over $\mathcal{H}$ that may depend on $X_{m+u}$ and let $\mathbf{q}$ be a (posterior) distribution over $\mathcal{H}$ that may depend on both $S_m$ and $X_u$. Let $\delta \in (0, 1)$ be given. With probability at least $1 - \delta$ over the choices of $S_m$ for any distribution $\mathbf{q}$*

$$
R_{G_\mathbf{q}}(X_u) \leq \hat{R}_{G_\mathbf{q}}(S_m) + \sqrt{\left( \frac{2\hat{R}_{G_\mathbf{q}}(S_m)(m+u)}{u} \right) \frac{D(\mathbf{q}\|\mathbf{p}) + \ln \frac{m}{\delta} + 7\log(m+u+1)}{m-1}}
$$
$$
+ \frac{2\left( D(\mathbf{q}\|\mathbf{p}) + \ln \frac{m}{\delta} + \frac{7}{m}\log(m+u+1) \right)}{m-1} \quad .
$$

In the context of inductive learning, a major obstacle in generating meaningful and effective bounds using the PAC-Bayesian framework [8] is the construction of "compact priors". Here we discuss two extensions to the PAC-Bayesian scheme, which together allow for easy choices of compact priors that can yield tight error bounds. The first extension we offer is the use of *multiple priors*. Instead of a single prior $\mathbf{p}$ in the original PAC-Bayesian framework we observe that one can use all PAC-Bayesian bounds with a number of priors $\mathbf{p}_1, \ldots, \mathbf{p}_k$ and then replace the complexity term $\ln(1/\mathbf{p}(h))$ (in Theorem 3.2) by $\min_i \ln(1/\mathbf{p}_i(h))$, at a cost of an additional $\ln k$ term (see below). Similarly, in Theorem 3.3 we can replace the KL-divergence term in the bound with $\min_i D(\mathbf{q}\|\mathbf{p}_i)$. The penalty for using $k$ priors is logarithmic in $k$ (specifically the $\ln(1/\delta)$ term in the original bound becomes $\ln(k/\delta)$). As long as $k$ is sub-exponential in $m$ we still obtain effective generalization bounds. The second "extension" is simply the feature of our transduction bounds (Theorems 3.2 and 3.3), which allows for the priors to be dependent on the full sample $X_{m+u}$. The combination of these two simple ideas yields a powerful technique for deriving error bounds in realistic transductive settings. After stating the extended result we later use it for deriving tight bounds for known learning algorithms and for deriving new algorithms. Suppose that instead of a single prior $\mathbf{p}$ over $\mathcal{H}$ we want to utilize $k$ priors, $\mathbf{p}_1, \ldots, \mathbf{p}_k$ and in retrospect choose the best among the $k$ corresponding PAC-Bayesian bounds. The following theorem shows that one can use polynomially many priors with a minor penalty. The proof, which is omitted due to space limitations, utilizes the union bound in a straightforward manner.

**Theorem 3.4** *Let the conditions of Theorem 3.2 hold, except that we now have $k$ prior distributions $\mathbf{p}_1, \ldots, \mathbf{p}_k$ defined over $\mathcal{H}$, each of which may depend on $X_{m+u}$. Let $\delta \in (0, 1)$ be given. Then, with probability at least $1 - \delta$ over random choices of sub-samples of size $m$ from the full-sample, for all $h \in \mathcal{H}$, (5) holds with $\mathbf{p}(h)$ replaced by $\min_{1 \leq i \leq k} \mathbf{p}_i(h)$ and $\log \frac{1}{\delta}$ is replaced by $\log \frac{k}{\delta}$.*

**Remark:** A similar result holds for the Gibbs algorithm of Theorem 3.3. Also, as noted by one of the reviewers, when the supports of the $k$ priors intersect (i.e. there is at least one pair of priors $\mathbf{p}_i$ and $\mathbf{p}_j$ with overlapping support), then one can do better by utilizing the "super prior" $\mathbf{p} = \frac{1}{k} \sum_i \mathbf{p}_i$ within the original Theorem 3.2. However, note that when the supports are disjoint, these two views (of multiple priors and a super prior) are equivalent. In the applications below we utilize non-intersecting priors.

## 4 Bounds for Compression Algorithms

Here we propose a technique for bounding the error of "compression" algorithms based on appropriate construction of prior probabilities. Let $\mathcal{A}$ be a learning algorithm. Intuitively, $\mathcal{A}$ is a "compression scheme" if it can generate the same hypothesis using a subset of the data. More formally, a learning algorithm $\mathcal{A}$ (viewed as a function from samples to some hypothesis class) is a compression scheme with respect to a sample $Z$ if there is a sub-sample $Z'$, $Z' \subset Z$, such that $\mathcal{A}(Z') = \mathcal{A}(Z)$. Observe that the SVM approach is a compression scheme, with $Z'$ being determined by the set of support vectors.

Let $\mathcal{A}$ be a deterministic compression scheme and consider the full sample $X_{m+u}$. For each integer $\tau = 1, \ldots, m$, consider all subsets of $X_{m+u}$ of size $\tau$, and for each subset construct all possible dichotomies of that subset (note that we are not proposing this approach as an algorithm, but rather as a means to derive bounds; in practice one need not construct all these dichotomies). A deterministic algorithm $\mathcal{A}$ uniquely determines at most one hypothesis $h \in \mathcal{H}$ for each dichotomy.[3] For each $\tau$, let the set of hypotheses generated by this procedure be denoted by $\mathcal{H}_\tau$. For the rest of this discussion we assume the worst case where $|H_\tau| = \binom{m+u}{\tau}$ (i.e. if $H_\tau$ does not contains one hypothesis for each dichotomy the bounds improve). The prior $\mathbf{p}_\tau$ is then defined to be a uniform distribution over $\mathcal{H}_\tau$. In this way we have $m$ priors, $\mathbf{p}_1, \ldots, \mathbf{p}_m$ which are constructed using only $X_{m+u}$ (and are independent of $S_m$). Any hypothesis selected by the learning algorithm $\mathcal{A}$ based on the labeled sample $S_m$ and on the test set $X_u$ belongs to $\cup_{\tau=1}^m \mathcal{H}_\tau$. The motivation for this construction is as follows. Each $\tau$ can be viewed as our "guess" for the maximal number of compression points that will be utilized by a resulting classifier. For each such $\tau$ the prior $\mathbf{p}_\tau$ is constructed over all possible classifiers that use $\tau$ compression points. By systematically considering all possible dichotomies of $\tau$ points we can characterize a relatively small subset of $\mathcal{H}$ without observing labels of the training points. Thus, each prior $\mathbf{p}_\tau$ represents one such guess. Using Theorem 3.4 we are later allowed to choose in retrospect the bound corresponding to the best "guess". The following corollary identifies an upper bound on the divergence in terms of the observed size of the compression set of the final classifier.

**Corollary 4.1** *Let the conditions of Theorem 3.4 hold. Let $\mathcal{A}$ be a deterministic learning algorithm leading to a hypothesis $h \in \mathcal{H}$ based on a compression set of size $s$. Then with probability at least $1 - \delta$ for all $h \in \mathcal{H}$, (5) holds with $\log(1/\mathbf{p}(h))$ replaced by $s \log(2e(m+u)/s)$ and $\ln(m/\delta)$ replaced by $\ln(m^2/\delta)$.*

**Proof:** Recall that $\mathcal{H}_s \subseteq \mathcal{H}$ is the support set of $\mathbf{p}_s$ and that $\mathbf{p}_s(h) = 1/|\mathcal{H}_s|$ for all $h \in \mathcal{H}_s$, implying that $\ln(1/\mathbf{p}_s(h)) = |\mathcal{H}_s|$. Using the inequality $\binom{m+u}{s} \leq (e(m+u)/s)^s$ we have that $|\mathcal{H}_s| = 2^s \binom{m+u}{s} \leq (2e(m+u)/s)^s$. Substituting this result in Theorem 3.4 while restricting the minimum over $i$ to be over $i \geq s$, leads to the desired result. $\square$

The bound of Corollary 4.1 can be easily computed once the classifier is trained. If the size of the compression set happens to be small, we obtain a tight bound. SVM classification is one of the best studied compression schemes. The compression set for a sample $S_m$ is given by the subset of support vectors. Thus the bound in Corollary 4.1 immediately applies with $s$ being the number of observed support vectors (after training). We note that this bound is similar to a recently derived compression bound for inductive learning (Theorem 5.18 in [16]). Also, observe that the algorithm itself (inductive SVM) did not use in this case the unlabeled sample (although the bound does use this sample). Nevertheless, using exactly the same technique we obtain error bounds for the *transductive* SVM algorithms in [2, 3].[4]

# 5 Bounds for Clustering Algorithms

Some learning problems do not allow for high compression rates using compression schemes such as SVMs (i.e. the number of support vectors can sometimes be very large). A considerably stronger type of compression can often be achieved by clustering algorithms. While there is lack of formal links between entirely unsupervised clustering and classification, within a transduction setting we can provide a principled approach to using clustering algorithms for classification. Let $\mathcal{A}$ be any (deterministic) clustering algorithm which, given the full sample $X_{m+u}$, can cluster this sample into any desired number of clusters. We use $\mathcal{A}$ to cluster $X_{m+u}$ into $2, 3 \ldots, c$ clusters where $c \leq m$. Thus, the algorithm generates a collection of partitions of $X_{m+u}$ into $\tau = 2, 3, \ldots, c$ clusters, where each partition is denoted by $C_\tau$. For each value of $\tau$, let $\mathcal{H}_\tau$ consist of those hypotheses which assign an identical label to all points in the same cluster of partition $C_\tau$, and define the prior $\mathbf{p}_\tau(h) = 1/2^\tau$ for each $h \in \mathcal{H}_\tau$ and zero otherwise (note that there are $2^\tau$ possible dichotomies). The learning algorithm selects a hypothesis as follows. Upon observing the labeled sample $S_m = (X_m, Y_m)$, for each of the clusterings $C_2, \ldots, C_c$ constructed above, it assigns a label to each cluster based on the majority vote from the labels $Y_m$ of points falling within the cluster (in case of ties, or if no points from $X_m$ belong to the cluster, choose a label arbitrarily). Doing this leads to $c - 1$ classifiers $h_\tau$, $\tau = 2, \ldots, c$. For each $h_\tau$ there is a valid error bound as given by Theorem 3.4 and all these bounds are valid simultaneously. Thus we choose the best classifier (equivalently, number of clusters) for which the best bound holds. We thus have the following corollary of Theorem 3.4 and Lemma 2.1.

**Corollary 5.1** *Let $\mathcal{A}$ be any clustering algorithm and let $h_\tau$, $\tau = 2, \ldots, c$ be classifications of test set $X_u$ as determined by clustering of the full sample $X_{m+u}$ (into $\tau$ clusters) and the training set $S_m$, as described above. Let $\delta \in (0, 1)$ be given. Then with probability at least $1 - \delta$, for all $\tau$, (5) holds with $\log(1/\mathbf{p}(h))$ replaced by $\tau$ and $\ln(m/\delta)$ replaced by $\ln(mc/\delta)$.*

Error bounds obtained using Corollary 5.1 can be rather tight when the clustering algorithm is successful (i.e. when it captures the class structure in the data using a small number of clusters).

Corollary 5.1 can be extended in a number of ways. One simple extension is the use of an *ensemble* of clustering algorithms. Specifically, we can concurrently apply $k$ clustering algorithm (using each algorithm to cluster the data into $\tau = 2, \ldots, c$ clusters). We thus obtain $kc$ hypotheses (partitions of $X_{m+u}$). By a simple application of the union bound we can replace $\ln \frac{cm}{\delta}$ by $\ln \frac{kcm}{\delta}$ in Corollary 5.1 and guarantee that $kc$ bounds hold simultaneously for all $kc$ hypotheses (with probability at least $1 - \delta$). We thus choose the hypothesis which minimizes the resulting bound. This extension is particularly attractive since typically without prior knowledge we do not know which clustering algorithm will be effective for the dataset at hand.

# 6 Concluding Remarks

We presented new bounds for transductive learning algorithms. We also developed a new technique for deriving tight error bounds for compression schemes and for clustering algorithms in the transductive setting. We expect that these bounds and new techniques will be useful for deriving new error bounds for other known algorithms and for deriving new types of transductive learning algorithms. It would be interesting to see if tighter transduction bounds can be obtained by reducing the "slacks" in the inequalities we use in our analysis. Another promising direction is the construction of better (multiple) priors. For example, in our compression bound (Corollary 4.1), for each number of compression points

we assigned the same prior to each possible point subset and each possible dichotomy. However, in practice a vast majority of all these subsets and dichotomies are unlikely to occur.

**Acknowledgments** The work of R.E and R.M. was partially supported by the Technion V.P.R. fund for the promotion of sponsored research. Support from the Ollendorff center of the department of Electrical Engineering at the Technion is also acknowledged. We also thank anonymous referees for their useful comments.

## Footnotes

[1]The original Setting 1, as proposed by Vapnik, discusses a full sample whose points are chosen independently at random according to some source distribution $\mu(x)$.

[3]It might be that for some dichotomies the algorithm will fail. For example, an SVM in feature space without soft margin will fail to classify non linearly-separable dichotomies of $X_{m+u}$.

[4]Note however that our bounds are optimized with a "minimum number of support vectors" approach rather than "maximum margin".

# References

[1] V. N. Vapnik. *Estimation of Dependences Based on Empirical Data*. Springer Verlag, New York, 1982.

[2] V. N. Vapnik. *Statistical Learning Theory*. Wiley Interscience, New York, 1998.

[3] T. Joachims. Transductive inference for text classification unsing support vector machines. In *European Conference on Machine Learning*, 1999.

[4] A. Blum and S. Chawla. Learning from labeled and unlabeled data using graph mincuts. In *Proceeding of The Eighteenth International Conference on Machine Learning (ICML 2001)*, pages 19–26, 2001.

[5] R. El-Yaniv and O. Souroujon. Iterative double clustering for unsupervised and semi-supervised learning. In *Advances in Neural Information Processing Systems (NIPS 2001)*, pages 1025–1032, 2001.

[6] T. Joachims. Transductive learning via spectral graph partitioning. In *Proceeding of The Twentieth International Conference on Machine Learning (ICML-2003)*, 2003.

[7] D. McAllester. Some PAC-Bayesian theorems. *Machine Learning*, 37(3):355–363, 1999.

[8] D. McAllester. PAC-Bayesian stochastic model selection. *Machine Learning*, 51(1):5–21, 2003.

[9] D. Wu, K. Bennett, N. Cristianini, and J. Shawe-Taylor. Large margin trees for induction and transduction. In *International Conference on Machine Learning*, 1999.

[10] L. Bottou, C. Cortes, and V. Vapnik. On the effective VC dimension. Technical report, AT&T, 1994.

[11] G.R.G. Lanckriet, N. Cristianini, L. El Ghaoui, P. Bartlett, and M.I. Jordan. Learning the kernel matrix with semi-definite programming. Technical report, University of Berkeley, Computer Science Division, 2002.

[12] A. Blum and J. Langford. Pac-mdl bounds. In *COLT*, pages 344–357, 2003.

[13] W. Hoeffding. Probability inequalities for sums of bounded random variables. *J. Amer. Statis. Assoc.*, 58:13–30, 1963.

[14] A. Dembo and O. Zeitouni. *Large Deviation Techniques and Applications*. Springer, New York, second edition, 1998.

[15] D. McAllester. Simplified pac-bayesian margin bounds. In *COLT*, pages 203–215, 2003.

[16] R. Herbrich. *Learning Kernel Classifiers: Theory and Algorithms*. MIT Press, Boston, 2002.
